# A hierarchical model of complex cells in visual cortex for the binocular perception of motion-in-depth

**Silvio P. Sabatini, Fabio Solari, Giulia Andreani,**
**Chiara Bartolozzi, and Giacomo M. Bisio**
Department of Biophysical and Electronic Engineering
University of Genoa, I-16145 Genova, ITALY
*silvio@dibe.unige.it*

## Abstract

A cortical model for motion-in-depth selectivity of complex cells in the visual cortex is proposed. The model is based on a time extension of the phase-based techniques for disparity estimation. We consider the computation of the total temporal derivative of the time-varying disparity through the combination of the responses of disparity energy units. To take into account the physiological plausibility, the model is based on the combinations of binocular cells characterized by different ocular dominance indices. The resulting cortical units of the model show a sharp selectivity for motion-in-depth that has been compared with that reported in the literature for real cortical cells.

## 1 Introduction

The analysis of a dynamic scene implies estimates of motion parameters to infer spatio-temporal information about the visual world. In particular, the perception of motion-in-depth (MID), i.e. the capability of discriminating between forward and backward movements of objects from an observer, has important implications for navigation in dynamic environments. In general, a reliable estimate of motion-in-depth can be gained by considering the dynamic stereo correspondence problem in the stereo image signals acquired by a binocular vision system. Fig. 1 shows the relationships between an object moving in the 3-D space and its geometrical projections in the right and left retinas. In a first approximation, the positions of corresponding points are related by a 1-D horizontal shift, the *disparity*, along the direction of the epipolar lines. Formally, the left and right observed intensities from the two eyes, respectively $I^L(x)$ and $I^R(x)$, result related as $I^L(x) = I^R[x + \delta(x)]$, where $\delta(x)$ is the horizontal binocular disparity. If an object moves from $P$ to $Q$ its disparity changes and projects different velocities $(v_L, v_R)$ on the retinas.

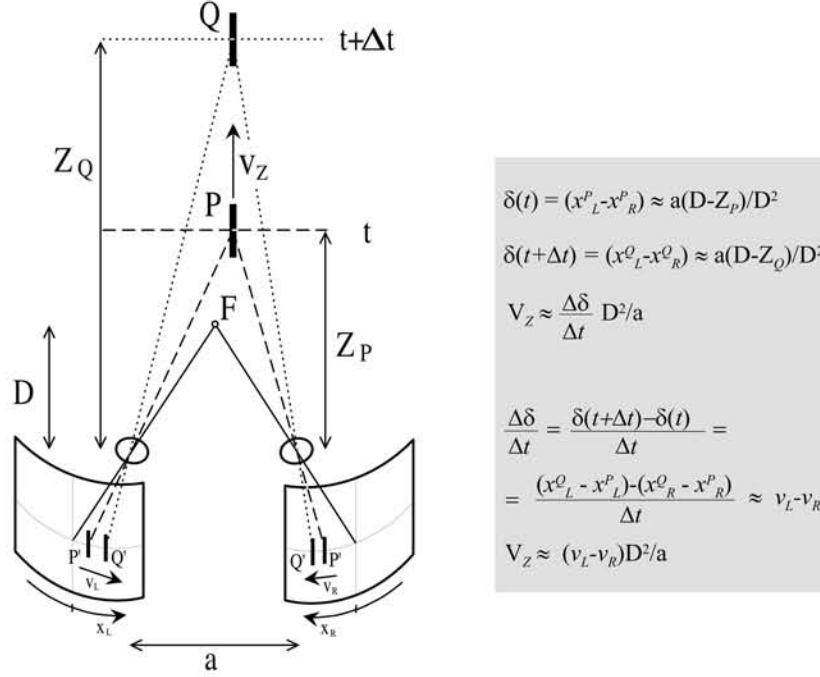

$$\delta(t) = (x^P{}_L - x^P{}_R) \approx a(D-Z_P)/D^2$$

$$\delta(t+\Delta t) = (x^Q{}_L - x^Q{}_R) \approx a(D-Z_Q)/D^2$$

$$V_Z \approx \frac{\Delta\delta}{\Delta t} D^2/a$$

$$\frac{\Delta\delta}{\Delta t} = \frac{\delta(t+\Delta t) - \delta(t)}{\Delta t} =$$

$$= \frac{(x^Q{}_L - x^P{}_L) - (x^Q{}_R - x^P{}_R)}{\Delta t} \approx v_L - v_R$$

$$V_Z \approx (v_L - v_R)D^2/a$$

Figure 1: The dynamic stereo correspondence problem. A moving object in the 3-D space projects different trajectories onto the left and right retinas. The differences between the two trajectories carry information about motion-in-depth.

Thus, the $Z$ component of the object's motion (i.e., its motion-in-depth) $V_Z$ can be approximated in two ways [1]: (1) by the rate of change of disparity, and (2) by the difference between retinal velocities, as it is evidenced in the box in Fig. 1. The predominance of one measure on the other one corresponds to different hypotheses on the architectural solutions adopted by visual cortical cells to encode dynamic 3-D visual information. Recently, numerous experimental and computational studies (see e.g., [2] [3] [4] [5]) addressed this issue, by analyzing the binocular spatio-temporal properties of simple and complex cells. The fact that the resulting disparity tuning does not vary with time, and that most of the cells in the primary visual cortex have the same motion preference for the two eyes, led to the conclusion that these cells are not tuned to motion-in-depth. In this paper, we demonstrate that, within a phase-based disparity encoding scheme, such cells relay phase temporal derivative components that can be combined, at a higher level, to yield a specific motion-in-depth selectivity. The rationale of this statement relies upon analytical considerations on phase-based dynamic stereopsis, as a time extension of the well-known phase-based techniques for disparity estimation [6] [7]. The resulting model is based on the computation of the total temporal derivative of the disparity through the combination of the outputs of binocular disparity energy units [4] [5] characterized by different ocular dominance indices. Since each energy unit is just a binocular Adelson and Bergen's motion detector, this establishes a link between the information contained in the total rate of change of the binocular

disparity and that held by the interocular velocity differences.

## 2 Phase-based dynamic stereopsis

In the last decades, a computational approach for stereopsis, that rely on the phase information contained in the spectral components of the stereo image pair, has been proposed [6] [7]. Spatially-localized phase measures on the left and right images can be obtained by filtering operations with a complex-valued quadrature pair of Gabor filters $h(x, k_0) = \mathrm{e}^{-x^2/\sigma^2}\mathrm{e}^{ik_0 x}$, where $k_0$ is the peak frequency of the filter and $\sigma$ relates to its spatial extension. The resulting convolutions with the left and right binocular signals can be expressed as $Q(x) = \rho(x)\mathrm{e}^{i\phi(x)} = C(x) + iS(x)$ where $\rho(x) = \sqrt{C^2(x) + S^2(x)}$ and $\phi(x) = \arctan(S(x)/C(x))$ denote their amplitude and phase components, respectively, and $C(x)$ and $S(x)$ are the responses of the quadrature pair of filters. Hence, binocular disparity can be predicted by $\delta(x) = [\phi^L(x) - \phi^R(x)]/k(x)$ where $k(x) = [\phi_x^L(x) + \phi_x^R(x)]/2$, with $\phi_x$ spatial derivative of phase $\phi$, is the average instantaneous frequency of the bandpass signal, that, under a linear phase model, can be approximated by the peak frequency of the Gabor filter $k_0$. Extending to time domain, the disparity of a point moving with the motion field can be estimated by:

$$\delta[x(t), t] = \frac{\phi^L[x(t), t] - \phi^R[x(t), t]}{k_0} \tag{1}$$

where phase components are computed from the spatiotemporal convolutions of the stereo image pair $Q(x, t) = C(x, t) + iS(x, t)$ with directionally tuned Gabor filters with a central frequency $\mathbf{p} = (k_0, \omega_0)$. For spatiotemporal locations where linear phase approximation still holds ($\phi \simeq k_0 x + \omega_0 t$), the phase differences in Eq. (1) provide only spatial information, useful for reliable disparity estimates.

### 2.1 Motion-in-depth

If disparity is defined with respect to the spatial coordinate $x_L$, by differentiating with respect to time, its total rate of variation can be written as

$$\frac{d\delta}{dt} = \frac{\partial\delta}{\partial t} + \frac{v_L}{k_0}\left(\phi_x^L - \phi_x^R\right) \tag{2}$$

where $v_L$ is the horizontal component of the velocity signal on the left retina. Considering the conservation property of local phase measurements [8], image velocities can be computed from the temporal evolution of constant phase contours, and thus:

$$\phi_x^L = -\frac{\phi_t^L}{v_L} \quad \text{and} \quad \phi_x^R = -\frac{\phi_t^R}{v_R} \tag{3}$$

with $\phi_t = \frac{\partial\phi}{\partial t}$. Combining Eq. (3) with Eq. (2) we obtain $d\delta/dt = (v_R - v_L)\phi_x^R/k_0$, where $(v_R - v_L)$ is the phase-based interocular velocity difference along the epipolar lines. When the spatial tuning frequency of the Gabor filter $k_0$ approaches the instantaneous spatial frequency of the left and right convolution signals one can derive the following approximated expressions:

$$\frac{d\delta}{dt} \simeq \frac{\partial\delta}{\partial t} = \frac{\phi_t^L - \phi_t^R}{k_0} \simeq v_R - v_L \tag{4}$$

The partial derivative of the disparity can be directly computed by convolutions $(S, C)$ of stereo image pairs and by their temporal derivatives $(S_t, C_t)$:

$$\frac{\partial \delta}{\partial t} = \left[ \frac{S_t^L C^L - S^L C_t^L}{(S^L)^2 + (C^L)^2} - \frac{S_t^R C^R - S^R C_t^R}{(S^R)^2 + (C^R)^2} \right] \frac{1}{k_0} \tag{5}$$

thus avoiding explicit calculation and differentiation of phase, and the attendant problem of phase unwrapping. Considering that, at first approximation $(S^L)^2 + (C^L)^2 \simeq (S^R)^2 + (C^R)^2$ and that these terms are scantly discriminant for motion-in-depth, we can formulate the cortical model taking into account the numerator terms only.

## 2.2   The cortical model

If one prefilters the image signal to extract some temporal frequency sub-band, $S(x, t) \simeq g * S(x, t)$ and $C(x, t) \simeq g * C(x, t)$, and evaluates the temporal changes in that sub-band, differentiation can be attained by convolutions on the data with appropriate bandpass temporal filters:

$$S'(x, t) \simeq g' * S(x, t) \quad ; \quad C'(x, t) \simeq g' * C(x, t) \ .$$

$S'$ and $C'$ approximate $S_t$ and $C_t$, respectively, if $g$ and $g'$ are a quadrature pair of temporal filters, e.g.: $g(t) = e^{-t/\tau} \sin \omega_0 t$ and $g'(t) = e^{-t/\tau} \cos \omega_0 t$. From a modeling perspective, that approximation allows us to express derivative operations in terms of convolutions with a set of spatio-temporal filters, whose shapes resemble those of simple cell receptive fields (RFs) of the primary visual cortex. Though, it is worthy to note that a direct interpretation of the computational model is not biologically plausible. Indeed, in the computational scheme (see Eq. (5)), the temporal variations of phases are obtained by processing monocular images separately and then the resulting signals are binocularly combined to give at an estimate of motion-in-depth in each spatial location. To employ binocular RFs from the beginning, as they exist for most of the cells in the visual cortex, we manipulated the numerator by rewriting it as the combination of terms characterized by a dominant contribution for the ipsilateral eye and a non-dominant contribution for the controlateral eye. These contributions are referable to binocular disparity energy units [5] built from two pairs of binocular direction selective simple cells with left and right RFs weighted by an ocular dominance index $\alpha \in [0, 1]$. The "tilted" spatio-temporal RFs of simple cells of the model are obtained by combining separable RFs according to an Adelson and Bergen's scheme [9]. It can be demonstrated that the information about motion-in-depth can be obtained with a minimum number of eight binocular simple cells, four with a left and four with a right ocular dominance, respectively (see Fig. 2):

$$S_1 = (1 - \alpha)(C_t^L + S^L) - \alpha(C^R - S_t^R) \quad ; \quad S_2 = (1 - \alpha)(C^L + S_t^L) + \alpha(C_t^R + S^R)$$

$$S_3 = (1 - \alpha)(C_t^L - S^L) - \alpha(C^R + S_t^R) \quad ; \quad S_4 = (1 - \alpha)(C^L + S_t^L) + \alpha(C_t^R - S^R)$$

$$S_5 = \alpha(C_t^L + S^L) - (1 - \alpha)(C^R - S_t^R) \quad ; \quad S_6 = \alpha(C^L - S_t^L) + (1 - \alpha)(C_t^R + S^R)$$

$$S_7 = \alpha(C_t^L - S^L) - (1 - \alpha)(C^R + S_t^R) \quad ; \quad S_8 = \alpha(C^L + S_t^L) + (1 - \alpha)(C_t^R - S^R)$$

$$C_{11} = S_1^2 + S_2^2 \quad ; \quad C_{12} = S_3^2 + S_4^2 \quad ; \quad C_{13} = S_5^2 + S_6^2 \quad ; \quad C_{14} = S_7^2 + S_8^2$$

$$C_{21} = C_{12} - C_{11} \quad ; \quad C_{22} = C_{13} - C_{14}$$
$$C_3 = (1 - 2\alpha)(S_t^L C^L - S^L C_t^L - S_t^R C^R + S^R C_t^R) \, .$$

The output of the higher complex cell in the hierarchy ($C_3$) truly encodes motion-in-depth information. It is worthy to note that for a balanced ocular dominance ($\alpha = 0.5$) the cell looses its selectivity.

## 3 Results

To assess model performances we derived cells' responses to drifting sinusoidal gratings with different speeds in the left and right eye. The spatial frequency of the gratings has been chosen as central to the RF's bandwidth. For each layer, the tuning characteristics of the cells are analyzed as sensitivity maps in the $(x_L - x_R)$ and $(v_L - v_R)$ domains for the static and dynamic properties, respectively. The $(x_L - x_R)$ represents the binocular RF [5] of a cell, evidencing its disparity tuning. The $(v_L - v_R)$ response represents the binocular tuning curve of the velocities along the epipolar lines. To better evidence motion-in-depth sensitivity, we represent as polar plots, the responses of the model cells with respect to the interocular velocities ratio for 12 different motion trajectories in depth (labeled 1 to 12) [10]. The cells of the cortical model exhibit properties and typical profiles similar to those observed in the visual cortex [5] [10]. The middle two layers (see insets A and B in Fig. 2) exhibit a strong selectivity to static disparity, but no specific tuning to motion-in-depth. On the contrary, the output cell $C_3$ shows a narrow tuning to the $Z$ direction of the object's motion, while lacking disparity tuning (see inset C in Fig. 2).

To consider more biologically plausible RFs for the simple cells, we included a coefficient $\beta$ in the scheme used to obtain tilted RFs in the space-time domain (e.g. $C + \beta S_t$). This coefficient takes into account the simple cell response to the non-preferred direction. We analytically demonstrated (results not shown here) that the resulting effect is a constant term that multiplies the cortical model output. In this way, the model is based on more realistic simple cells without lacking its functionality, provided that the basic direction selective units maintain a significant direction selective index. To analyze the effect of the architectural parameters on the model performance, we systematically varied the ocular dominance index $\alpha$ and introduced a weight $\gamma$ representing the inhibition strength of the afferent signals to the complex cells in layer 2. The resulting direction-in-depth polar plots are shown in Fig. 3. The $\alpha$ parameter yields a strong effect on the response profile: if $\alpha = 0.5$ there is no direction-in-depth selectivity; according that $\alpha > 0.5$ or $\alpha < 0.5$ cells exhibit a tuning to opposite directions in depth. As $\alpha$ approaches the boundary values 0 or 1 the binocular model turns to a monocular one. A decrease of the inhibition strength $\gamma$ yields cells characterized by a less selective response to direction-in-depth, whereas an increase of $\gamma$ diminishes their response amplitude.

## 4 Discussion and conclusions

There are at least two binocular cues that can be used to determine the MID [1]: binocular combination of monocular velocity signals or the rate of change of retinal disparity. Assuming a phase-based disparity encoding scheme [6], we demonstrated that information held in the interocular velocity difference is the same of

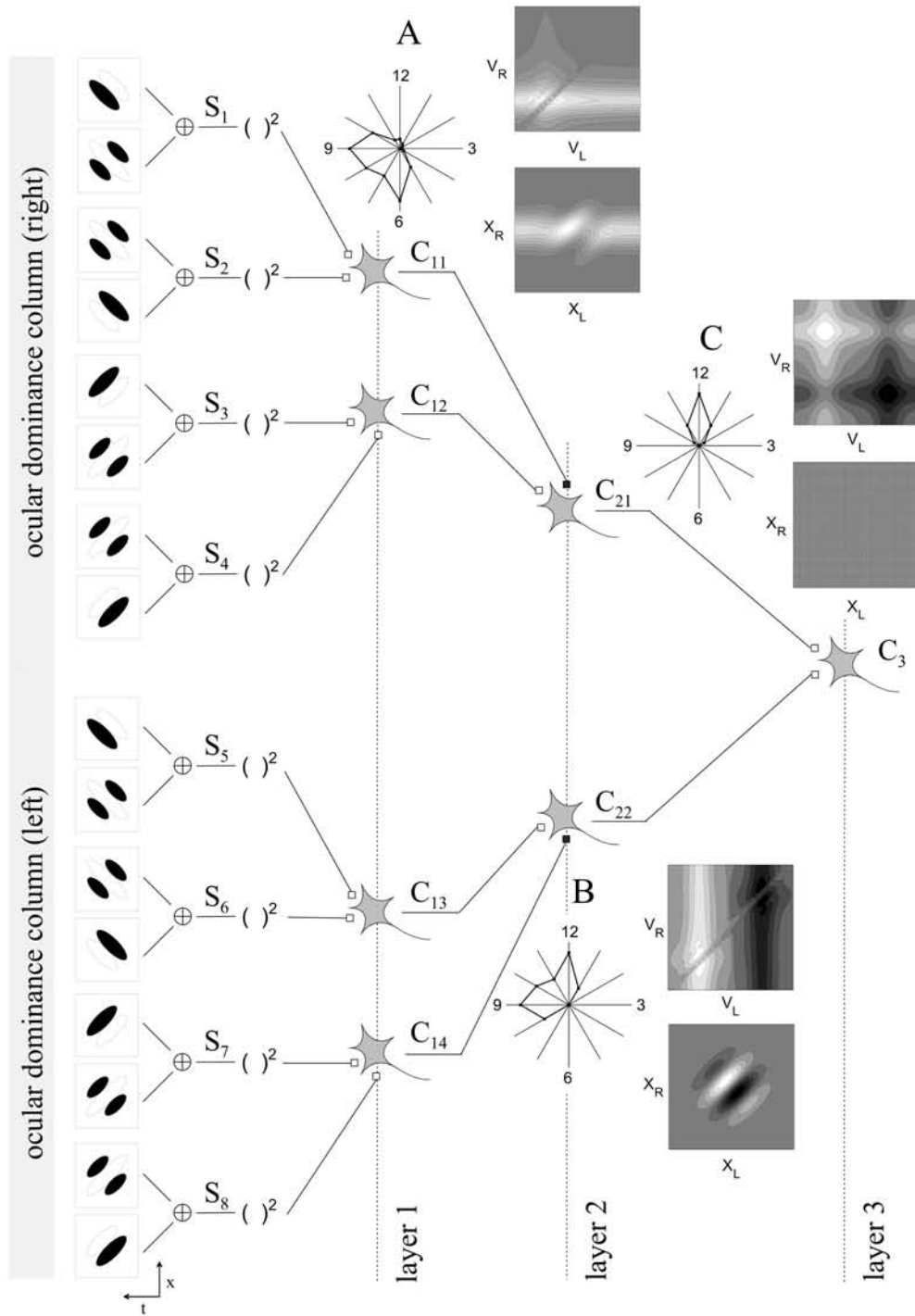

Figure 2: Functional representation of the proposed cortical architecture. Each branch groups cells belonging to an ocular dominance column. The afferent signals from left and right ocular dominance columns are combined in layer 3. The basic units are binocular simple cells tuned to motion directions ($S_1, \ldots, S_8$). The responses of the complex cells in layers 1, 2 and 3 are obtained by linear and nonlinear combinations of the outputs of those basic units. See text. White squares denote excitatory synapses whereas black squares denote inhibitory ones.

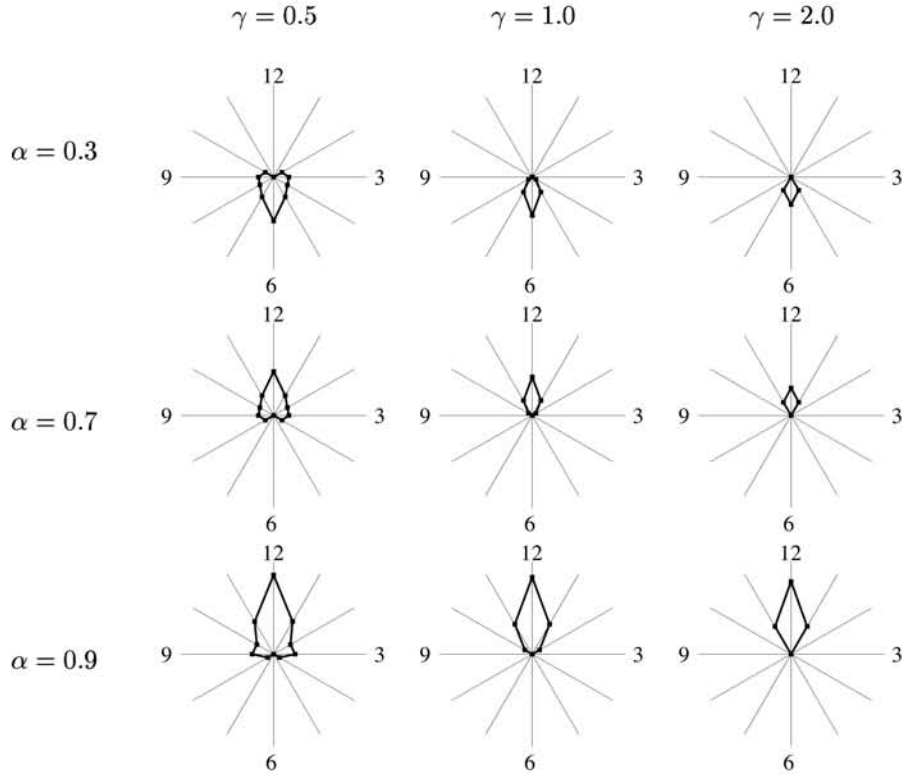

Figure 3: Effects on the direction-in-depth selectivity of the systematic variation of the model's parameters $\alpha$ and $\gamma$. The responses are normalized to the largest amplitude value.

that derived by the evaluation of the total derivative of the binocular disparity. The resulting computation relies upon spatio-temporal differentials of the left and right retinal phases that can be approximated by linear filtering operations with spatio-temporal RFs. Accordingly, we proposed a cortical model for the generation of binocular motion-in-depth selective cells as a hierarchical combination of binocular energy complex cells. It is worth noting that the phase response and the associated characteristic disparity of simple and complex cells in layers 1 and 2 do not change with time, but the amplitudes of their responses carry information on temporal phase derivatives, that can be related to both retinal velocities and temporal changes in disparity. Moreover, the model evidences the different roles of simple and complex cells. Simple cells provide a Gabor-like spatio-temporal transformation of the visual space, on which to base a variety of visual functions (perception of form, depth, motion). Complex cells, by proper combinations of the same signals provided by simple cells, actively eliminate sensitivity to a selected set of parameters, thus becoming specifically tuned to different features, such as disparity but not motion-in-depth (layer 1 and 2), motion-in-depth but not disparity (layer 3).

**Acknowledgments**

This work was partially supported by the *UNIGE-2000 Project "Spatio-temporal Operators for the Analysis of Motion in Depth from Binocular Images"*.

# References

[1] J. Harris and S. N.J. Watamaniuk. Speed discrimination of Motion-in depth using binocular cues. *Vision Research*, 35(7):885–896, 1995.

[2] N. Qian and S. Mikaelian. Relationship between phase and energy methods for disparity computation. *Neural Comp.*, 12(2):279–292, 2000.

[3] Y. Chen, Y. Wang, and N. Qian. Modelling V1 disparity tuning to time-varying stimuli. *J. Neurophysiol.*, pages 504–600, 2001.

[4] D. J. Fleet, H. Wagner, and D. J. Heeger. Neural encoding of binocular diparity: energy models, position shift and phase shift. *Vision Research*, 17:345–398, 1996.

[5] I. Ohzawa, G.C. DeAngelis, and R.D. Freeman. Encoding of binocular disparity by complex cells in the cat's visual cortex. *J. Neurophysiol.*, 77:2879–2909, 1997.

[6] T.D. Sanger. Stereo disparity computation using Gabor filters. *Biol. Cybern.*, 59:405–418, 1988.

[7] D.J. Fleet, A.D. Jepson, and M. Jenkin. Phase-based disparity measurements. *CVGIP: Image Understanding*, 53:198–210, 1991.

[8] D. J. Fleet and A. D. Jepson. Computation of component image velocity from local phase information. *International Journal of Computer Vision*, 1:77–104, 1990.

[9] E.H. Adelson and J.R. Bergen. Spatiotemporal energy models for the perception of motion. *J. Opt. Soc. Amer.*, 2:284–321, 1985.

[10] W. Spileers, G.A. Orban, B. Gulyàs, and H. Maes. Selectivity of cat area 18 neurons for direction and speed in depth. *J. Neurophysiol.*, 63(4):936–954, 1990.
